# An Architectural Mechanism for Direction-tuned Cortical Simple Cells: The Role of Mutual Inhibition

**Silvio P. Sabatini**
silvio@dibe.unige.it

**Fabio Solari**
fabio@dibe.unige.it

**Giacomo M. Bisio**
bisio@dibe.unige.it

Department of Biophysical and Electronic Engineering
PSPC Research Group
Genova, I-16145, Italy

## Abstract

A linear architectural model of cortical simple cells is presented. The model evidences how mutual inhibition, occurring through synaptic coupling functions asymmetrically distributed in space, can be a possible basis for a wide variety of spatio-temporal simple cell response properties, including direction selectivity and velocity tuning. While spatial asymmetries are included explicitly in the structure of the inhibitory interconnections, temporal asymmetries originate from the specific mutual inhibition scheme considered. Extensive simulations supporting the model are reported.

## 1 INTRODUCTION

One of the most distinctive features of striate cortex neurons is their combined selectivity for stimulus orientation and the direction of motion. The majority of simple cells, indeed, responds better to sinusoidal gratings that are moving in one direction than to the opposite one, exhibiting also a narrower velocity tuning with respect to that of geniculate cells. Recent theoretical and neurophysiological studies [1] [2] pointed out that the initial stage of direction selectivity can be related to the linear space-time receptive field structure of simple cells. A large class of simple cells has a very specific space-time behavior in which the spatial phase of the receptive field changes gradually as a function of time. This results in receptive field profiles that are tilted in the space-time domain. To account for the origin of this particular spatio-temporal inseparability, numerous models have been proposed postulating the existence of structural asymmetries of the geniculo-cortical projections both in the temporal and in the spatial domains (for a review, see [3]

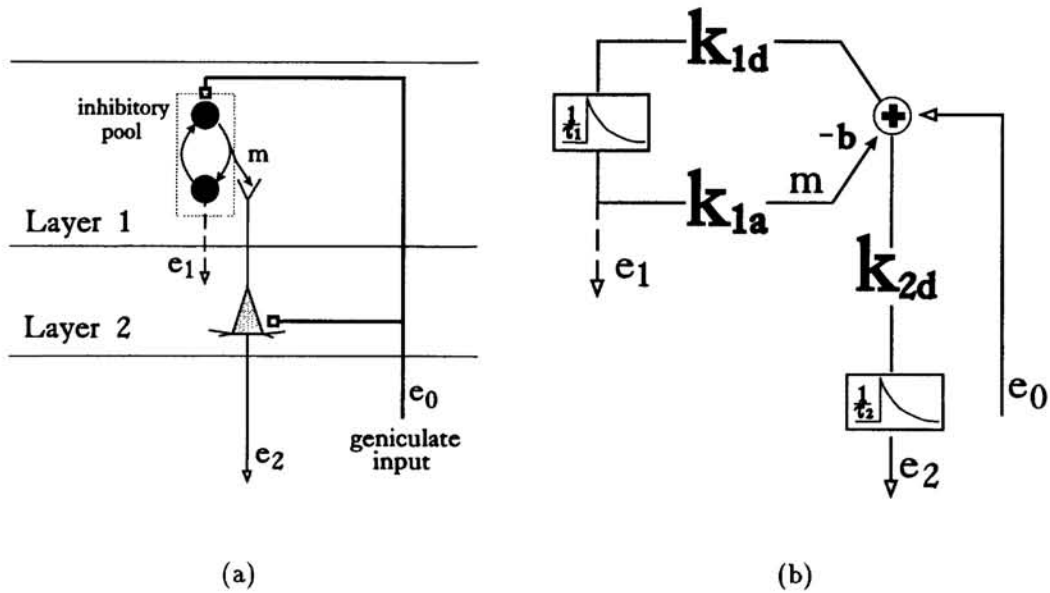

Figure 1: (a) A schematic neural circuitry for the mutual inhibition; (b) equivalent block diagram representation.

[4]). Among them, feed-forward inhibition along the non-preferred direction, and the combination of lagged and non-lagged geniculate inputs to the cortex have been commonly suggested as the major mechanisms.

In this paper, within a linear field theory framework, we propose and analyse an architectural model for dynamic receptive field formation, based on intracortical interactions occurring through asymmetric mutual inhibition schemes.

## 2  MODELING INTRACORTICAL PROCESSING

The computational characteristics of each neuron are not independent of the ones of other neurons laying in the same layer, rather, they are often the consequence of a collective behavior of neighboring cells. To understand how intracortical circuits may affect the response properties of simple cells one can study their structure and function at many levels of organization, from subcellular, driven primarily by biophysical data, to systemic, driven by functional considerations. In this study, we present a model at the intermediate abstraction level to combine both functional and neurophysiological descriptions into an analytic model of cortical simple cells.

### 2.1  STRUCTURE OF THE MODEL

Following a linear neural field approach [5] [6], we regard visual cortex as a continuous distribution of neurons and synapses. Accordingly, the geniculo-cortical pathway is modeled by a multi-layer network interconnected through feed-forward and feedback connections, both inter- and intra-layers. Each location on the cortical plane represents a homogeneous population of cells, and connections represent average interactions among populations. Such connections can be modeled by spatial coupling functions which represent the spread of the synaptic influence of a

population on its neighbors, as mediated by local axonal and dendritic fields. From an architectural point of view, we assume the superposition of feed-forward (i.e., geniculate) and intracortical contributions which arise from inhibitory pools whose activity is also primed by a geniculate excitatory drive. A schematic diagram showing the "building blocks" of the model is depicted in Fig. 1. The dynamics of each population is modeled as first-order low-pass filters characterized by time constants $\tau$'s. For the sake of simplicity, we restrict our analysis to 1-D case, assuming that such direction is orthogonal to the preferred direction of the receptive field [7]. This 1-D model would produce spatio-temporal results that are directly compared with the spatio-temporal plots usually obtained when an optimal stimulus is moved along the direction orthogonal to the preferred direction of the receptive field.

Geniculate contributions $e_0(x, t)$ are modeled directly by a spatiotemporal convolution of the visual input $s(x, t)$ and a separable kernel $h_0(x, t)$ characterized in the spatial domain by a Gaussian shape with spatial extent $\sigma_0$ and, in the temporal domain, by a first-order temporal low-pass filter with time constant $\tau_0$. The output $e_1(x, t)$ of the inhibitory neuron population results from the mutual inhibitory scheme through spatially organized pre- and post-synaptic sites, modeled by the kernels $k_{1a}(x - \xi)$ and $k_{1d}(x - \xi)$, respectively:

$$\tau_1 \frac{de_1(x, t)}{dt} = -e_1(x, t) + \int k_{1d}(x - \xi)[e_0(-\xi, t) - bm(-\xi, t)]d\xi \qquad (1)$$

$$m(x, t) = \int k_{1a}(x - \xi)e_1(-\xi)d\xi \qquad (2)$$

where the function $m(x, t)$ describes the spatio-temporal mutual inhibitory interactions, and $b$ is the inhibition strength. The layer 2 cortical excitation $e_2(x, t)$ is the result of feed-forward contributions collected ($k_{2d}$) from the inhibitory loop, at axonal synaptic sites, and the geniculate input ($e_0(x, t)$). To focus the attention on the inhibitory loop, in the following we assume a one-to-one mapping from layer 1 to layer 2, i.e., $k_{2d}(x - \xi) = \delta(x - \xi)$, consequently:

$$\tau_2 \frac{de_2(x, t)}{dt} = -e_2(x, t) + e_0(x, t) - bm(x, t) \qquad (3)$$

where $\tau_1$ and $\tau_2$ are the time constants associated to layer 1 and layer 2, respectively.

## 2.2 AVERAGE INTRACORTICAL CONNECTIVITY

When assessing the role of intracortical circuits on the receptive field properties of cortical cells, one important issue concerns the spatial localization of inhibitory and excitatory influences. In a previous work [8] we evidenced how the steady-state solution of Eqs. (1)-(3) can give rise to highly structured Gabor-like receptive field profiles, when inhibition arises from laterally distributed clusters of cells. In this case, the effective intrinsic kernel $k_1(x)$, defined as $k_1(x - \xi) \stackrel{\text{def}}{=} \int \int k_{1a}(-x', -\xi')k_{1d}(x - x', \xi - \xi')dx'd\xi'$, can be modeled as the sum of two Gaussians symmetrically offset with respect to the target cell (see Fig. 2):

$$k_1(x) = \frac{1}{\sqrt{2\pi}} \left( \frac{w_1}{\sigma_1} \exp[-(x - d_1)^2/2\sigma_1^2] + \frac{w_2}{\sigma_2} \exp[-(x + d_2)^2/2\sigma_2^2] \right). \qquad (4)$$

This work is aimed to investigate how spatial asymmetries in the intracortical coupling function lead to non-separable space-time interactions within the resulting discharge field of the simple cells. To this end, we varied systematically the geometrical parameters $(\sigma, w, d)$ of the inhibitory kernel to consider three different

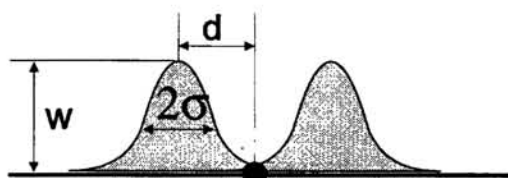

Figure 2: The basic inhibitory kernel used $k_1(x - \xi)$. The cell in the center receives inhibitory contributions from laterally distributed clusters of cells. The asymmetric kernels used in the model derive from this basic kernel by systematic variations of its geometrical parameters (see Table 1).

types of asymmetries: (1) different *spatial spread* of inhibition (i.e., $\sigma_1 \neq \sigma_2$); (2) different *amount* of inhibition ($w_1 \neq w_2$); (3) different *spatial offset* ($d_1 \neq d_2$). A more rigorous treatment should take care also of the continuous distortion of the topographic map [9]. In our analysis this would result in a continuous deformation of the inhibitory kernel, but for the small distances within which inhibition occurs, the approximation of a uniform mapping produces only a negligible error.

Architectural parameters were determined from reliable measured values of receptive fields of simple cells [10] [11]. Concerning the spatial domain, we fixed the size ($\sigma_0$) of the initial receptive field (due to geniculate contributions) for an "average" cortical simple cell with a resultant discharge field of $\sim 2^o$; and we adjusted, accordingly, the parameters of the inhibitory kernel in order to account for spatial interactions only within the receptive field.

Considering the temporal domain, one should distinguish the time constant $\tau_1$, caused by network interactions, from the time constants $\tau_0$ and $\tau_2$ caused by temporal integration at a single cell membrane. In any case, throughout all simulations, we fixed $\tau_0$ and $\tau_2$ to 20ms, whereas we varied $\tau_1$ in the range 2 - 100ms.

## 3  RESULTS

Since visual cortex is visuotopically organized, a direct correspondence exists between the spatial organization of intracortical connections and the resulting receptive field topography. Therefore, the dependence of cortical surface activity $e_2(x, t)$ on the visual input $s(x, t)$ can be formulated as $e_2(x, t) = h(x, t) * s(x, t)$, where the symbol $*$ indicates a spatio-temporal convolution, and $h(x, t)$ is the equivalent receptive field interpreted as the spatio-temporal distribution of the signs of all the effects of cortical interactions. In this context, $h(x, t)$ reflects the whole spatio-temporal couplings and not only the direct neuroanatomical connectivity.

To test the efficacy of the various inhibitory schemes, we used a drifting sine wave grating $s(x, t) = C \cos[2\pi(f_x x \pm f_t t)]$ where $C$ is the contrast, $f_x$ and $f_t$ are the spatial and temporal frequency, respectively. The direction selectivity index (DSI) and the optimal velocity ($v_{opt}$) obtained from the various inhibitory kernels of Fig.2 are summarized in Table 1, for different values of $\tau_1$ and $b$. The direction selectivity index is defined as $DSI = \frac{R_p - R_{np}}{R_p + R_{np}}$, where $R_p$ is the maximum response amplitude for preferred direction, and $R_{np}$ is the maximum amplitude for non-preferred direction. The optimal velocity is defined as $f_t^{opt}/f_x$, where $f_x$ is chosen to match the spatial structure of the receptive field, and $f_t^{opt}$ is the frequency which elicits the maximum cell's response.   As expected, increasing the parameter $b$ enhances the effects of inhibition, thus resulting in larger DSI and higher optimal velocities. However, for stability reason, $b$ should remain below a theshold value strictly re-

Table 1:

| b | $\tau_1 = 2$ DSI | $v_{opt}$ | $\tau_1 = 10$ DSI | $v_{opt}$ | $\tau_1 = 20$ DSI | $v_{opt}$ | $\tau_1 = 100$ DSI | $v_{opt}$ | |
|---|---|---|---|---|---|---|---|---|---|
| 0.25 | 0.00 | 0.00 | 0.00 | 0.00 | 0.00 | 0.00 | 0.00 | 0.00 | |
| 0.60 | 0.00 | 0.00 | 0.00 | 0.00 | 0.00 | 0.00 | 0.00 | 0.00 | ASY-1A 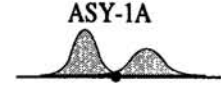 |
| 0.80 | 0.08 | 1.82 | 0.24 | 1.82 | 0.00 | 0.00 | 0.00 | 0.00 | |
| 0.91 | 0.17 | 1.82 | **0.34** | **1.82** | 0.00 | 0.00 | 0.00 | 0.00 | |
| 0.50 | 0.00 | 0.00 | 0.00 | 0.00 | 0.00 | 0.00 | 0.00 | 0.00 | |
| 0.85 | 0.04 | 1.82 | 0.17 | 1.82 | 0.25 | 1.82 | 0.00 | 0.00 | ASY-1B 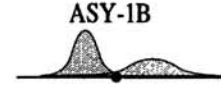 |
| 1.00 | 0.06 | 1.82 | 0.19 | 1.82 | 0.28 | 1.82 | 0.00 | 0.00 | |
| 1.30 | 0.07 | 1.82 | 0.28 | 1.82 | **0.37** | **1.82** | 0.00 | 0.00 | |
| 0.50 | 0.00 | 0.00 | 0.00 | 0.00 | 0.00 | 0.00 | 0.00 | 0.00 | |
| 1.00 | 0.00 | 0.00 | 0.00 | 0.00 | 0.09 | 1.82 | 0.16 | 1.82 | ASY-2A 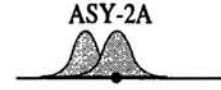 |
| 5.00 | 0.02 | 1.82 | 0.05 | 1.82 | 0.09 | 1.82 | **0.39** | **3.64** | |
| 9.00 | 0.01 | 1.82 | 0.03 | 1.82 | 0.06 | 1.82 | 0.38 | 3.64 | |
| 0.25 | 0.00 | 0.00 | 0.00 | 0.00 | 0.00 | 0.00 | 0.00 | 0.00 | |
| 0.50 | 0.06 | 2.07 | 0.20 | 2.07 | 0.32 | 2.07 | 0.00 | 0.00 | ASY-2B 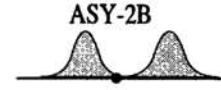 |
| 0.60 | 0.06 | 2.07 | 0.39 | 4.14 | 0.40 | 2.07 | 0.00 | 0.00 | |
| 0.72 | 0.07 | 2.00 | **0.66** | **6.00** | 0.65 | 4.00 | 0.00 | 0.00 | |
| 0.25 | 0.00 | 0.00 | 0.00 | 0.00 | 0.00 | 0.00 | 0.00 | 0.00 | |
| 0.60 | 0.00 | 00.0 | 0.00 | 0.00 | 0.00 | 0.00 | 0.00 | 0.00 | ASY-3A 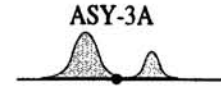 |
| 0.80 | 0.08 | 1.82 | 0.23 | 1.82 | 0.00 | 0.00 | 0.00 | 0.00 | |
| 0.88 | 0.14 | 1.82 | **0.26** | **1.82** | 0.00 | 0.00 | 0.00 | 0.00 | |
| 0.50 | 0.00 | 0.00 | 0.00 | 0.00 | 0.00 | 0.00 | 0.00 | 0.00 | |
| 0.85 | 0.04 | 1.82 | 0.16 | 1.82 | 0.23 | 1.82 | 0.00 | 0.00 | ASY-3B 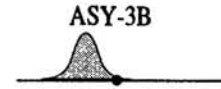 |
| 1.00 | 0.05 | 1.82 | 0.18 | 1.82 | 0.26 | 1.82 | 0.00 | 0.00 | |
| 1.33 | 0.06 | 1.82 | 0.26 | 1.82 | **0.35** | **1.82** | 0.00 | 0.00 | |

lated to the inhibitory kernel considered. Moreover, we observe that, except for ASY-2A, the strongest direction selectivity can be obtained when the intracortical time constant $\tau_1$ has values in the range of 10 - 20 ms, i.e., comparable to $\tau_0$ and $\tau_2$. Larger values of $\tau_1$ would result, indeed, in a recrudescence of the velocity low-pass behavior. For each asymmetry, Figs. 3 show the direction tuning curves and the x-t plots, respectively, for the best cases considered (cf. bold-faced values in Table 1). We have evidenced that appreciable DSI can be obtained when inhibition arises from cortical sites at different distance from the target cell (i.e., ASY-2B, $d_1 \neq d_2$). In such conditions we obtained a DSI as high as 0.66 and an optimal velocity up to $\sim 6°/s$, as could be inferred also from the spatio-temporal plot which present a marked motion-type (i.e., oriented) non-separability (see Fig. 3ASY-2B).

## 4   DISCUSSION AND CONCLUSIONS

As anticipated in the Introduction, direction selectivity mechanisms usually relies upon asymmetric alteration of the spatial and temporal response characteristics of the geniculate input, which are presumably mediated by intracortical circuits. In the architectural model presented in this study, spatial asymmetries were included explicitly in the extension of the inhibitory interconnections, but no explicit asymmetric temporal mechanisms were introduced. It is worth evidencing how temporal asymmetries originate from the specific mutual inhibition scheme considered, which operates, regarding temporal domain, like a quadrature model [12] [13]. This can

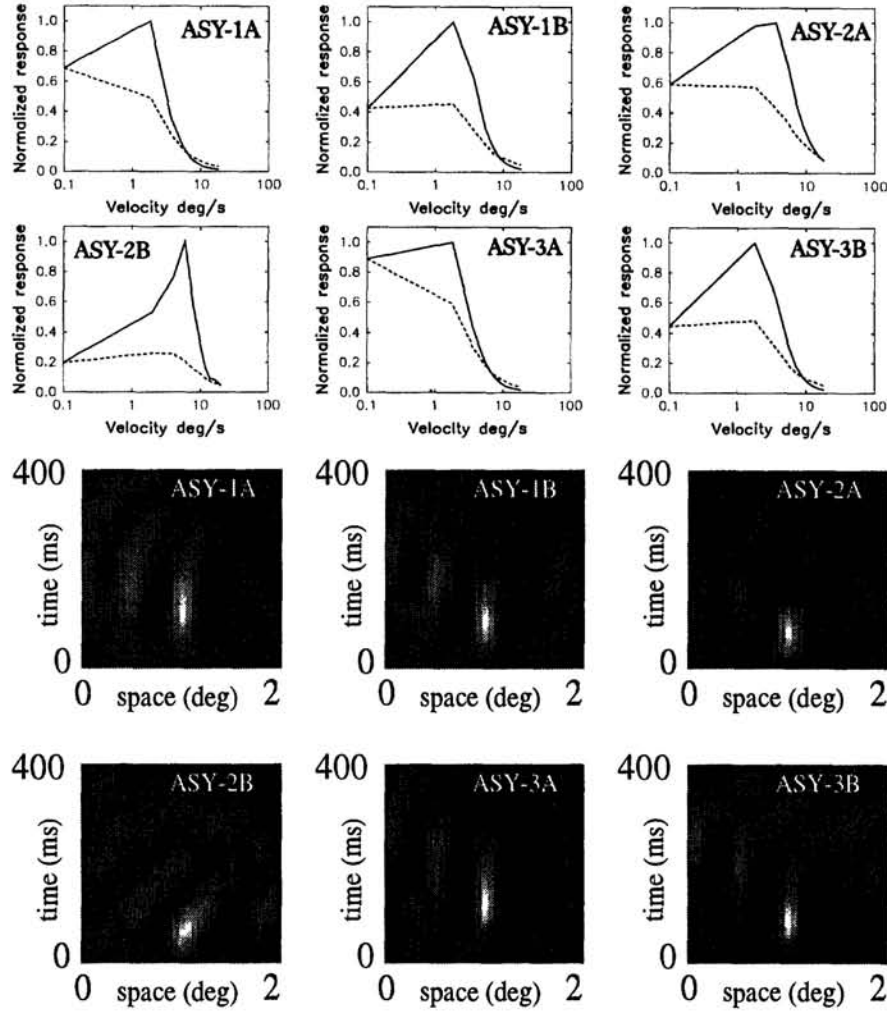

Figure 3: Results from the model related to the bold-typed values indicated in Table 1, for each asymmetry considered. (Top) direction tuning curve; (Bottom) spatio-temporal plots. We can evidence a marked direction tuning for ASY-2B, i.e., when inhibition arises from two differentially offset Gaussians

be inferred by the examination of the equivalent transfer function $H(\omega_x, \omega_t)$ in the Fourier domain:

$$H(\omega_x, \omega_t) = \frac{H_0(\omega_x, \omega_t)}{1 + j\omega_t \tau_2} \left( \frac{1}{1 + j\omega_t \tau_1 + bK_1(\omega_x)} + j\omega_t \tau_1 \frac{1}{1 + j\omega_t \tau_1 + bK_1(\omega_x)} \right)$$
(5)

where upper case letters indicate Fourier transforms, and $j$ is the complex variable. The terms in parentheses in Eq. (5) can be interpreted as the sum of temporal components that are approximately arranged in temporal quadrature. Furthermore, one can observe that a direct monosynaptic influence ($e_1$) from the inhibitory neurons of layer 1 to the excitatory cells of layer 2, would result in the cancellation of the quadrature component in Eq. (5).

Further improvement of the model should take into account also transmission delays between spatially separated interacting cells, theshold non-linearities, and ON-OFF interactions.

**Acknowledgements**

This research was partially supported by CEC-Esprit CORMORANT 8503, and by MURST 40%-60%.

# References

[1] R.C. Reid, R.E. Soodak, and R.M. Shapley. Directional selectivity and spatiotemporal structure of receptive fields of simple cells in cat striate cortex. *J. Neurophysiol.*, 66:505–529, 1991.

[2] D.B. Hamilton, D.G. Albrecht, and W.S. Geisler. Visual cortical receptive fields in monkey and cat: spatial and temporal phase transfer function. *Vision Res.*, 29(10):1285–1308, 1989.

[3] K. Nakayama. Biological image motion processing: a review. *Vision Res.*, 25:625–660, 1985.

[4] E.C. Hildreth and C. Koch. The analysis of visual motion: From computational theory to neuronal mechanisms. *Ann. Rev. Neurosci.*, 10:477–533, 1987.

[5] G. Krone, H. Mallot, G. Palm, and A. Schüz. Spatiotemporal receptive fields: A dynamical model derived from cortical architectonics. *Proc. R. Soc. London Biol*, 226:421–444, 1986.

[6] H.R. Wilson and J.D. Cowan. A mathematical theory of the functional dynamics of cortical and thalamic nervous tissue. *Kibernetik*, 13:55–80, 1973.

[7] G.C. De Angelis, I. Ohzawa, and R.D. Freeman. Spatiotemporal organization of simple-cell receptive fields in the cat's striate cortex.I. General characteristics and postnatal development. *J. Neurophysiol.*, 69:1091–1117, 1993.

[8] S.P. Sabatini, L. Raffo, and G.M. Bisio. Functional periodic intracortical couplings induced by structured lateral inhibition in a linear cortical network. *Neural Computation*, 9(3):525–531, 1997.

[9] H.A. Mallot, W. von Seelen, and F. Giannakopoulos. Neural mapping and space variant image processing. *Neural Networks*, 3:245–263, 1990.

[10] K. Albus. A quantitative study of the projection area of the central and the paracentral visual field in area 17 of the cat. *Exp. Brain Res.*, 24:159–202, 1975.

[11] J. Jones and L. Palmer. The two-dimensional spatial structure of simple receptive fields in cat striate cortex. *J. Neurophysiol.*, 58:1187–1211, 1987.

[12] A.B. Watson and A.J. Ahumada. Model of human visual-motion sensing. *J. Opt. Soc. Amer.*, 2:322–341, 1985.

[13] E.H. Adelson and J.R. Bergen. Spatiotemporal energy models for the perception of motion. *J. Opt. Soc. Amer.*, 2:284–321, 1985.
